# Large-Scale Sparsified Manifold Regularization

**Ivor W. Tsang**       **James T. Kwok**
Department of Computer Science and Engineering
The Hong Kong University of Science and Technology
Clear Water Bay, Kowloon, Hong Kong
{ivor,jamesk}@cse.ust.hk

## Abstract

Semi-supervised learning is more powerful than supervised learning by using both labeled and unlabeled data. In particular, the manifold regularization framework, together with kernel methods, leads to the Laplacian SVM (LapSVM) that has demonstrated state-of-the-art performance. However, the LapSVM solution typically involves kernel expansions of all the labeled and unlabeled examples, and is slow on testing. Moreover, existing semi-supervised learning methods, including the LapSVM, can only handle a small number of unlabeled examples. In this paper, we integrate manifold regularization with the core vector machine, which has been used for large-scale supervised and unsupervised learning. By using a sparsified manifold regularizer and formulating as a center-constrained minimum enclosing ball problem, the proposed method produces sparse solutions with low time and space complexities. Experimental results show that it is much faster than the LapSVM, and can handle a million unlabeled examples on a standard PC; while the LapSVM can only handle several thousand patterns.

## 1 Introduction

In many real-world applications, collection of labeled data is both time-consuming and expensive. On the other hand, a large amount of unlabeled data are often readily available. While traditional supervised learning methods can only learn from the limited amount of labeled data, semi-supervised learning [2] aims at improving the generalization performance by utilizing both the labeled and unlabeled data. The label dependencies among patterns are captured by exploiting the intrinsic geometric structure of the data. The underlying smoothness assumption is that two nearby patterns in a high-density region should share similar labels [2]. When the data lie on a manifold, it is common to approximate this manifold by a weighted graph, leading to graph-based semi-supervised learning methods. However, many of these are designed for transductive learning, and thus cannot be easily extended to out-of-sample patterns.

Recently, attention is drawn to the development of inductive methods, such as harmonic mixtures [15] and Nyström-based methods [3]. In this paper, we focus on the manifold regularization framework proposed in [1]. By defining a data-dependent reproducing kernel Hilbert space (RKHS), manifold regularization incorporates an additional regularizer to ensure that the learned function is smooth on the manifold. Kernel methods, which have been highly successful in supervised learning, can then be integrated with this RKHS. The resultant Laplacian SVM (LapSVM) demonstrates state-of-the-art semi-supervised learning performance [10]. However, a deficiency of the LapSVM is that its solution, unlike that of the SVM, is not sparse and so is much slower on testing.

Moreover, while the original motivation of semi-supervised learning is to utilize the large amount of unlabeled data available, existing algorithms are only capable of handling a small to moderate amount of unlabeled data. Recently, attempts have been made to scale up these methods. Sindhwani *et al.* [9] speeded up manifold regularization by restraining to linear models, which, however, may

not be flexible enough for complicated target functions. Garcke and Griebel [5] proposed to use discretization with a sparse grid. Though it scales linearly with the sample size, its time complexity grows exponentially with data dimensionality. As reported in a recent survey [14], most semi-supervised learning methods can only handle 100 – 10,000 unlabeled examples. More recently, Gärtner *et al.* [6] presented a solution in the more restrictive transductive setting. The largest graph they worked with involve 75,888 labeled and unlabeled examples. Thus, no one has ever been experimented on massive data sets with, say, one million unlabeled examples.

On the other hand, the Core Vector Machine (CVM) is recently proposed for scaling up kernel methods in both supervised (including classification [12] and regression [13]) and unsupervised learning (e.g., novelty detection). Its main idea is to formulate the learning problem as a minimum enclosing ball (MEB) problem in computational geometry, and then use an $(1 + \epsilon)$-approximation algorithm to obtain a close-to-optimal solution efficiently. Given $m$ samples, the CVM has an asymptotic time complexity that is only *linear* in $m$ and a space complexity that is even *independent* of $m$ for a fixed $\epsilon$. Experimental results on real world data sets with millions of patterns demonstrated that the CVM is much faster than existing SVM implementations and can handle much larger data sets.

In this paper, we extend the CVM to semi-supervised learning. To restore sparsity of the LapSVM solution, we first introduce a sparsified manifold regularizer based on the $\epsilon$-insensitive loss. Then, we incorporate manifold regularization into the CVM. It turns out that the resultant QP can be casted as a center-constrained MEB problem introduced in [13]. The rest of this paper is organized as follows. In Section 2, we first give a brief review on manifold regularization. Section 3 then describes the proposed algorithm for semi-supervised classification and regression. Experimental results on very large data sets are presented in Section 4, and the last section gives some concluding remarks.

## 2 Manifold Regularization

Given a training set $\{(\mathbf{x}_i, y_i)\}_{i=1}^m$ with input $\mathbf{x}_i \in \mathcal{X}$ and output $y_i \in \mathbb{R}$. The *regularized risk functional* is the sum of the empirical risk (corresponding to a loss function $\ell$) and a regularizer $\Omega$. Given a kernel $k$ and its RKHS $\mathcal{H}_k$, we minimize the regularized risk over function $f$ in $\mathcal{H}_k$:

$$\min_{f \in \mathcal{H}_k} \frac{1}{m} \sum_{i=1}^m \ell(\mathbf{x}_i, y_i, f(\mathbf{x}_i)) + \lambda \Omega(\|f\|_{\mathcal{H}_k}). \tag{1}$$

Here, $\|\cdot\|_{\mathcal{H}_k}$ denotes the RKHS norm and $\lambda > 0$ is a regularization parameter. By the representer theorem, the minimizer $f$ admits the representation $f(\mathbf{x}) = \sum_{i=1}^m \alpha_i k(\mathbf{x}_i, \mathbf{x})$, where $\alpha_i \in \mathbb{R}$. Therefore, the problem is reduced to the optimization over the finite-dimensional space of $\alpha_i$'s.

In semi-supervised learning, we have both labeled examples $\{(\mathbf{x}_i, y_i)\}_{i=1}^m$ and unlabeled examples $\{\mathbf{x}_i\}_{i=m+1}^{m+n}$. Manifold regularization uses an additional regularizer $\|f\|_{\mathcal{I}}^2$ to ensure that the function $f$ is smooth on the intrinsic structure of the input. The objective function in (1) is then modified to:

$$\frac{1}{m} \sum_{i=1}^m \ell(\mathbf{x}_i, y_i, f(\mathbf{x}_i)) + \lambda \Omega(\|f\|_{\mathcal{H}_k}) + \lambda_{\mathcal{I}} \|f\|_{\mathcal{I}}^2, \tag{2}$$

where $\lambda_{\mathcal{I}}$ is another tradeoff parameter. It can be shown that the minimizer $f$ is of the form $f(\mathbf{x}) = \sum_{i=1}^m \alpha_i k(\mathbf{x}_i, \mathbf{x}) + \int_{\mathcal{M}} \alpha(\mathbf{x}') k(\mathbf{x}, \mathbf{x}') dP_{\mathcal{X}}(\mathbf{x}')$, where $\mathcal{M}$ is the support of the marginal distribution $P_{\mathcal{X}}$ of $\mathcal{X}$ [1].

In practice, we do not have access to $P_{\mathcal{X}}$. Now, assume that the support $\mathcal{M}$ of $P_{\mathcal{X}}$ is a compact sub-manifold, and take $\|f\|_{\mathcal{I}}^2 = \int_{\mathcal{M}} \langle \nabla f, \nabla f \rangle$ where $\nabla$ is the gradient of $f$. It is common to approximate this manifold by a weighted graph defined on all the labeled and unlabeled data, as $G = (\mathcal{V}, \mathcal{E})$ with $\mathcal{V}$ and $\mathcal{E}$ being the sets of vertices and edges respectively. Denote the weight function $w$ and degree $d(u) = \sum_{v \sim u} w(u, v)$. Here, $v \sim u$ means that $u, v$ are adjacent. Then, $\|f\|_{\mathcal{I}}^2$ is approximated as[1]

$$\|f\|_{\mathcal{I}}^2 = \sum_{e \in \mathcal{E}} \left| \sqrt{w(u_e, v_e)} \left( \frac{f(\mathbf{x}_{u_e})}{s(u_e)} - \frac{f(\mathbf{x}_{v_e})}{s(v_e)} \right) \right|^2, \tag{3}$$

where $u_e$ and $v_e$ are vertices of the edge $e$, and $s(u) = \sqrt{d(u)}$ when the normalized graph Laplacian is used, and $s(u) = 1$ with the unnormalized one. As shown in [1], the minimizer of (2) becomes $f(\mathbf{x}) = \sum_{i=1}^{m+n} \alpha_i k(\mathbf{x}_i, \mathbf{x})$, which depends on both labeled and unlabeled examples.

## 2.1 Laplacian SVM

Using the hinge loss $\ell(\mathbf{x}_i, y_i, f(\mathbf{x}_i)) = \max(0, 1 - y_i f(\mathbf{x}_i))$ in (2), we obtain the Laplacian SVM (LapSVM) [1]. Its training involves two steps. First, solve the quadratic program (QP) $\max \ \boldsymbol{\beta}'\mathbf{1} - \frac{1}{2}\boldsymbol{\beta}'\mathbf{Q}\boldsymbol{\beta} \ : \ \boldsymbol{\beta}'\mathbf{y} = 0, \ \mathbf{0} \le \boldsymbol{\beta} \le \frac{1}{m}\mathbf{1}$ to obtain the $\boldsymbol{\beta}^*$. Here, $\boldsymbol{\beta} = [\beta_1, \ldots, \beta_m]', \mathbf{1} = [1, \ldots, 1]', \mathbf{Q}_{m \times m} = \mathbf{YJK}(2\lambda\mathbf{I} + 2\lambda_{\mathcal{I}}\mathbf{LK})^{-1}\mathbf{J}'\mathbf{Y}$, $\mathbf{Y}_{m \times m}$ is the diagonal matrix with $\mathbf{Y}_{ii} = y_i$, $\mathbf{K}_{(m+n) \times (m+n)}$ is the kernel matrix over both the labeled and unlabeled data, $\mathbf{L}_{(m+n) \times (m+n)}$ is the graph Laplacian, and $\mathbf{J}_{m \times (m+n)}$ with $\mathbf{J}_{ij} = 1$ if $i = j$ and $\mathbf{x}_i$ is a labeled example, and $\mathbf{J}_{ij} = 0$ otherwise. The optimal $\boldsymbol{\alpha} = [\alpha_1, \ldots, \alpha_{m+n}]'$ solution is then obtained by solving the linear system: $\boldsymbol{\alpha}^* = (2\lambda\mathbf{I} + 2\lambda_{\mathcal{I}}\mathbf{LK})^{-1}\mathbf{J}'\mathbf{Y}\boldsymbol{\beta}^*$. Note that the matrix $2\lambda\mathbf{I} + 2\lambda_{\mathcal{I}}\mathbf{LK}$ is of size $(m+n) \times (m+n)$, and so its inversion can be very expensive when $n$ is large. Moreover, unlike the standard SVM, the $\boldsymbol{\alpha}^*$ obtained is not sparse and so evaluation of $f(x)$ is slow.

# 3 Proposed Algorithm

## 3.1 Sparsified Manifold Regularizer

To restore sparsity of the LapSVM solution, we replace the square function in the manifold regularizer (3) by the $\epsilon$-insensitive loss function[2], as

$$\|f\|_{\mathcal{I}}^2 = \sum_{e \in \mathcal{E}} \left| \sqrt{w(u_e, v_e)} \left( \frac{f(\mathbf{x}_{u_e})}{s(u_e)} - \frac{f(\mathbf{x}_{v_e})}{s(v_e)} \right) \right|_{\bar{\varepsilon}}^2, \tag{4}$$

where $|z|_{\bar{\varepsilon}} = 0$ if $|z| \le \bar{\varepsilon}$; and $|z| - \bar{\varepsilon}$ otherwise. Obviously, it reduces to (3) when $\bar{\varepsilon} = 0$. As will be shown in Section 3.3, the $\boldsymbol{\alpha}$ solution obtained will be sparse. Substituting (4) into (2), we have:

$$\min_{f \in \mathcal{H}_k} \left\{ \frac{1}{m} \sum_{i=1}^m \ell(\mathbf{x}_i, y_i, f(\mathbf{x}_i)) + \lambda_{\mathcal{I}} \sum_{e \in \mathcal{E}} \left| \sqrt{w(u_e, v_e)} \left( \frac{f(\mathbf{x}_{u_e})}{s(u_e)} - \frac{f(\mathbf{x}_{v_e})}{s(v_e)} \right) \right|_{\bar{\varepsilon}}^2 \right\} + \lambda\Omega(\|f\|_{\mathcal{H}_k}).$$

By treating the terms inside the braces as the "loss function", this can be regarded as regularized risk minimization and, using the standard representer theorem, the minimizer $f$ then admits the form $f(\mathbf{x}) = \sum_{i=1}^{m+n} \alpha_i k(\mathbf{x}_i, \mathbf{x})$, same as that of the original manifold regularization.

Moreover, putting $f(\mathbf{x}) = \mathbf{w}'\varphi(\mathbf{x}) + b$ into (4), we obtain $\|f\|_{\mathcal{I}}^2 = \sum_{e \in \mathcal{E}} |\mathbf{w}'\boldsymbol{\psi}_e + b\tau_e|_{\bar{\varepsilon}}^2$, where $\boldsymbol{\psi}_e = \sqrt{w(u_e, v_e)} \left( \frac{\varphi(\mathbf{x}_{u_e})}{s(u_e)} - \frac{\varphi(\mathbf{x}_{v_e})}{s(v_e)} \right)$, and $\tau_e = \sqrt{w(u_e, v_e)} \left( \frac{1}{s(u_e)} - \frac{1}{s(v_e)} \right)$. The primal of the LapSVM can then be formulated as:

$$\min \quad \|\mathbf{w}\|^2 + b^2 + \frac{C}{m\mu} \sum_{i=1}^m \xi_i^2 + 2C\bar{\varepsilon} + \frac{C\theta}{|\mathcal{E}|\mu} \sum_{e \in \mathcal{E}} (\zeta_e^2 + \zeta_e^{*2}) \tag{5}$$

$$\text{s.t.} \quad y_i(\mathbf{w}'\varphi(\mathbf{x}_i) + b) \ge 1 - \bar{\varepsilon} - \xi_i, \quad i = 1, \ldots, m, \tag{6}$$

$$-(\mathbf{w}'\boldsymbol{\psi}_e + b\tau_e) \le \bar{\varepsilon} + \zeta_e, \quad \mathbf{w}'\boldsymbol{\psi}_e + b\tau_e \le \bar{\varepsilon} + \zeta_e^*, \quad e \in \mathcal{E}. \tag{7}$$

Here, $|\mathcal{E}|$ is the number of edges in the graph, $\xi_i$ is the slack variable for the error, $\zeta_e, \zeta_e^*$ are slack variables for edge $e$, and $C, \mu, \theta$ are user-defined parameters. As in previous CVM formulations [12, 13], the bias $b$ is penalized and the two-norm errors ($\xi_i^2, \zeta_{ij}^2$ and $\zeta_{ij}^{*2}$) are used. Moreover, the constraints $\xi_i, \zeta_{ij}, \zeta_{ij}^*, \bar{\varepsilon} \ge 0$ are automatically satisfied. When $\bar{\varepsilon} = 0$, (5) reduces to the original LapSVM (using two-norm errors). When $\theta$ is also zero, it becomes the Lagrangian SVM.

The dual can be easily obtained as the following QP:

$$\max \ [\boldsymbol{\beta}' \ \boldsymbol{\gamma}' \ \boldsymbol{\gamma}^{*'}][\tfrac{2}{C}\mathbf{1}' \ \mathbf{0}' \ \mathbf{0}']' - [\boldsymbol{\beta}' \ \boldsymbol{\gamma}' \ \boldsymbol{\gamma}^{*'}]\tilde{\mathbf{K}}[\boldsymbol{\beta}' \ \boldsymbol{\gamma}' \ \boldsymbol{\gamma}^{*'}]' \ : \ [\boldsymbol{\beta}' \ \boldsymbol{\gamma}' \ \boldsymbol{\gamma}^{*'}]\mathbf{1} = 1, \boldsymbol{\beta}, \boldsymbol{\gamma}, \boldsymbol{\gamma}^* \ge \mathbf{0}, \tag{8}$$

where $\boldsymbol{\beta} = [\beta_1, \ldots, \beta_m]'$, $\boldsymbol{\gamma} = [\gamma_1, \ldots, \gamma_{|\mathcal{E}|}]'$, $\boldsymbol{\gamma}^* = [\gamma_1^*, \ldots, \gamma_{|\mathcal{E}|}^*]'$ are the dual variables, and $\tilde{\mathbf{K}} =$

$$\begin{bmatrix} (\mathbf{K}_\ell + \mathbf{1}\mathbf{1}' + \frac{\mu m}{C}\mathbf{I}) \odot \mathbf{y}\mathbf{y}' & \mathbf{V} & -\mathbf{V} \\ \mathbf{V}' & \mathbf{U} + \frac{|\mathcal{E}|\mu}{C\theta}\mathbf{I} & -\mathbf{U} \\ -\mathbf{V}' & -\mathbf{U} & \mathbf{U} + \frac{|\mathcal{E}|\mu}{C\theta}\mathbf{I} \end{bmatrix}$$ is the transformed "kernel matrix". Here,

$\mathbf{K}_\ell$ is the kernel matrix defined using kernel $k$ on the $m$ labeled examples, $\mathbf{U}_{|\mathcal{E}| \times |\mathcal{E}|} = [\boldsymbol{\psi}_e' \boldsymbol{\psi}_f + \tau_e \tau_f]$, and $\mathbf{V}_{m \times |\mathcal{E}|} = [y_i \varphi(\mathbf{x}_i)' \boldsymbol{\psi}_e + \tau_e]$. Note that while each entry of the matrix $\mathbf{Q}$ in LapSVM (Section 2.1) requires $O((m + n)^2)$ kernel $k(\mathbf{x}_i, \mathbf{x}_j)$ evaluations, each entry in $\tilde{\mathbf{K}}$ here takes only $O(1)$ kernel evaluations. This is particularly favorable to decomposition methods such as SMO as most of the CPU computations are typically dominated by kernel evaluations.

Moreover, it can be shown that $\mu$ is a parameter that controls the size of $\bar{\varepsilon}$, analogous to the $\nu$ parameter in $\nu$-SVR. Hence, only $\mu$, but not $\bar{\varepsilon}$, appears in (8). Moreover, the primal variables can be easily recovered from the dual variables by the KKT conditions. In particular, $\mathbf{w} = C \left( \sum_{i=1}^m \beta_i y_i \varphi(\mathbf{x}_i) + \sum_{e \in \mathcal{E}} (\gamma_e - \gamma_e^*) \boldsymbol{\psi}_e \right)$ and $b = C \left( \sum_{i=1}^m \beta_i y_i + \sum_{e \in \mathcal{E}} (\gamma_e - \gamma_e^*) \tau_e \right)$. Subsequently, the decision function $f(\mathbf{x}) = \mathbf{w}' \varphi(\mathbf{x}) + b$ is a linear combination of $k(\mathbf{x}_i, \mathbf{x})$'s defined on both the labeled and unlabeled examples, as in standard manifold regularization.

### 3.2 Transforming to a MEB Problem

We now show that CVM can be used for solving the possibly very large QP in (8). In particular, we will transform this QP to the dual of a *center-constrained MEB problem* [13], which is of the form:

$$\max \; \boldsymbol{\alpha}'(\mathrm{diag}(\mathbf{K}) + \boldsymbol{\Delta} - \eta\mathbf{1}) - \boldsymbol{\alpha}'\mathbf{K}\boldsymbol{\alpha} \; : \; \boldsymbol{\alpha} \geq \mathbf{0}, \; \boldsymbol{\alpha}'\mathbf{1} = 1, \qquad (9)$$

for some $\mathbf{0} \leq \boldsymbol{\Delta} \in \mathbb{R}^m$ and $\eta \in \mathbb{R}$. From the variables in (8), define $\tilde{\boldsymbol{\alpha}} = [\boldsymbol{\beta}' \; \boldsymbol{\gamma}' \; \boldsymbol{\gamma}^{*'}]'$ and $\boldsymbol{\Delta} = -\mathrm{diag}(\tilde{\mathbf{K}}) + \eta\mathbf{1} + \frac{2}{C}[\mathbf{1}' \; \mathbf{0}' \; \mathbf{0}]'$ s.t. $\boldsymbol{\Delta} \geq \mathbf{0}$ for some sufficiently large $\eta$. (8) can then be written as $\max \; \tilde{\boldsymbol{\alpha}}'(\mathrm{diag}(\tilde{\mathbf{K}}) + \boldsymbol{\Delta} - \eta\mathbf{1}) - \tilde{\boldsymbol{\alpha}}'\tilde{\mathbf{K}}\tilde{\boldsymbol{\alpha}} \; : \; \tilde{\boldsymbol{\alpha}} \geq \mathbf{0}, \tilde{\boldsymbol{\alpha}}'\mathbf{1} = 1$, which is of the form in (9).

The above formulation can be easily extended to the regression case, with the pattern output changed from $\pm 1$ to $y_i \in \mathbb{R}$, and the hinge loss replaced by the $\epsilon$-insensitive loss. Converting the resultant QP to the form in (9) is also straightforward.

### 3.3 Sparsity

In Section 3.3.1, we first explain why a sparse solution can be obtained by using the KKT conditions. Alternatively, by building on [7], we show in Section 3.3.2 that the $\epsilon$-insensitive loss achieves a similar effect as the $\ell_1$ penalty in LASSO [11], which is known to produce sparse approximation.

#### 3.3.1 KKT Perspective

Basically, this follows from the standard argument as for sparse solutions with the $\epsilon$-insensitive loss in SVR. From the KKT condition associated with (6): $\beta_i(y_i(\mathbf{w}'\varphi(\mathbf{x}_i) + b) - 1 + \bar{\varepsilon} + \xi_i) = 0$. As for the SVM, most patterns are expected to lie outside the margin (i.e. $y_i(\mathbf{w}'\varphi(\mathbf{x}_i) + b) > 1 - \bar{\varepsilon}$) and so most $\beta_i$'s are zero. Similarly, manifold regularization finds a $f$ that is locally smooth. Hence, from the definition of $\boldsymbol{\psi}_e$ and $\tau_e$, many values of $(\mathbf{w}'\boldsymbol{\psi}_e + b\tau_e)$'s will be inside the $\bar{\varepsilon}$-tube. Using the KKT conditions associated with (7), the corresponding $\gamma_e$'s and $\gamma_e^*$'s are zero. As $f(\mathbf{x})$ is a linear combination of the $k(\mathbf{x}_i, \mathbf{x})$'s weighted by $\beta_i$ and $\gamma_e - \gamma_e^*$ (Section 3.1), $f$ is thus sparse.

#### 3.3.2 LASSO Perspective

Our exposition will be along the line pioneered by Girosi [7], who established a connection between the $\epsilon$-insensitive loss in SVR and sparse approximation. Given a predictor $f(\mathbf{x}) = \sum_{i=1}^m \alpha_i k(\mathbf{x}_i, \mathbf{x}) = \mathbf{K}\boldsymbol{\alpha}$, we consider minimizing the error between $\mathbf{f} = [f(\mathbf{x}_1), \ldots f(\mathbf{x}_m)]'$ and $\mathbf{y} = [y_1, \ldots, y_m]'$. While sparse approximation techniques such as basis pursuit typically use the $L_2$ norm for the error, Girosi argued that the norm of the RKHS $\mathcal{H}_k$ is a better measure of smoothness. However, the RKHS norm operates on functions, while here we have vectors $\mathbf{f}$ and $\mathbf{y}$ w.r.t. $\mathbf{x}_1, \ldots, \mathbf{x}_m$. Hence, we will use the kernel PCA map with $\|\mathbf{y} - \mathbf{f}\|_{\mathbf{K}}^2 \equiv (\mathbf{y} - \mathbf{f})\mathbf{K}^{-1}(\mathbf{y} - \mathbf{f})$.

First, consider the simpler case where the manifold regularizer is replaced by a simple regularizer $\|\alpha\|_2^2$. As in LASSO, we also add a $\ell_1$ penalty on $\alpha$. The optimization problem is formulated as:

$$\min \ \|\mathbf{y} - \mathbf{f}\|_{\mathbf{K}}^2 + \frac{\mu m}{C}\alpha'\alpha \ : \ \|\alpha\|_1 = C, \tag{10}$$

where $C$ and $\mu$ are constants. As in [7], we decompose $\alpha$ as $\beta - \beta^*$, where $\beta, \beta^* \geq \mathbf{0}$ and $\beta_i \beta_i^* = 0$. Then, (10) can be rewritten as:

$$\max \ [\beta' \ \beta^{*\prime}][2\mathbf{y}' \ -2\mathbf{y}']' - [\beta' \ \beta^{*\prime}]\tilde{\mathbf{K}}[\beta' \ \beta^{*\prime}]' \ : \ \beta, \beta^* \geq \mathbf{0}, \ \beta'\mathbf{1} + \beta^{*\prime}\mathbf{1} = C, \tag{11}$$

where[3] $\tilde{\mathbf{K}} = \begin{bmatrix} \mathbf{K} + \frac{\mu m}{C}\mathbf{I} & -\mathbf{K} \\ -\mathbf{K} & \mathbf{K} + \frac{\mu m}{C}\mathbf{I} \end{bmatrix}$. On the other hand, consider the following variant of SVR using the $\epsilon$-insensitive loss:

$$\min \|\mathbf{w}\|^2 + \frac{C}{m\mu}\sum_{i=1}^{m}(\xi_i^2 + \xi_i^{*2}) + 2C\bar{\varepsilon} \ : \ y_i - \mathbf{w}'\varphi(\mathbf{x}_i) \leq \bar{\varepsilon} + \xi_i, \ \mathbf{w}'\varphi(\mathbf{x}_i) - y_i \leq \bar{\varepsilon} + \xi_i^*. \tag{12}$$

It can be shown that its dual is identical to (11), with $\beta, \beta^*$ as dual variables. Moreover, the LASSO penalty (i.e., the equality constraint in (11)) is induced from the $\bar{\varepsilon}$ in (12). Hence, the $\epsilon$-insensitive loss in SVR achieves a similar effect as using the error $\|\mathbf{y} - \mathbf{f}\|_{\mathbf{K}}^2$ and the LASSO penalty.

We now add back the manifold regularizer. The derivation is similar, though more involved, and so details are skipped. As above, the key steps are on replacing the $\ell_2$ norm by the kernel PCA map, and adding a $\ell_1$ penalty on the variables. It can then be shown that sparsified manifold regularizer (based on the $\epsilon$-insensitive loss) can again be recovered by using the LASSO penalty.

### 3.4 Complexities

As the proposed algorithm is an extension of the CVM, its properties are analogous to those in [12]. For example, its approximation ratio is $(1 + \epsilon)^2$, and so the approximate solution obtained is very close to the exact optimal solution. As for the computational complexities, it can be shown that the SLapCVM only takes $O(1/\epsilon^8)$ time and $O(1/\epsilon^2)$ space when probabilistic speedup is used. (Here, we ignore $O(m + |\mathcal{E}|)$ space required for storing the $m$ training patterns and $2|\mathcal{E}|$ edge constraints, as these may be stored outside the core memory.) They are thus independent of the numbers of labeled and unlabeled examples for a fixed $\epsilon$. In contrary, LapSVM involves an expensive matrix inversion for $\mathbf{K}_{(m+n)\times(m+n)}$ and requires $O((m + n)^3)$ time and $O((m + n)^2)$ space.

### 3.5 Remarks

The reduced SVM [8] has been used to scale up the standard SVM. Hence, another natural alternative is to extend it for the LapSVM. This "reduced LapSVM" solves a smaller optimization problem that involves a random $r \times (m+n)$ rectangular subset of the kernel matrix, where the $r$ patterns are chosen from both the labeled and unlabeled data. It can be easily shown that it requires $O((m + n)^2 r)$ time and $O((m + n)r)$ space. Experimental comparisons based on this will be made in Section 4.

Note that the CVM [12] is in many aspects similar to the column generation technique [4] commonly used in large-scale linear or integer programs. Both start with only a small number of nonzero variables, and the restricted master problem in column generation corresponds to the inner QP that is solved at each CVM iteration. Moreover, both can be regarded as primal methods that maintain primal[4] feasibility and work towards dual feasibility. Also, as is typical in column generation, the dual variable whose KKT condition is most violated is added at each iteration. The key difference[5],

however, is that CVM exploits the "approximateness" as in other approximation algorithms. Instead of requiring the dual solution to be strictly feasible, CVM only requires it to be feasible within a factor of $(1 + \epsilon)$. This, together with the fact that its dual is a MEB problem, allows its number of iterations for convergence to be bounded and thus the total time and space complexities guaranteed. On the other hand, we are not aware of any similar results for column generation.

By regarding the CVM as the approximation algorithm counterpart of column generation, this suggests that the CVM can also be used in the same way as column generation in speeding up other optimization problems. For example, the CVM can also be used for SVM training with other loss functions (e.g. 1-norm error). However, as the dual may no longer be a MEB problem, the downside is that its convergence bound and complexity results in Section 3.4 may no longer be available.

## 4 Experiments

In this section, we perform experiments on some massive data sets [6] (Table 1). The graph (for the manifold) is constructed by using the 6 nearest neighbors of each pattern, and the weight $w(u_e, v_e)$ in (3) is defined as $\exp(-\|\mathbf{x}_{u_e} - \mathbf{x}_{v_e}\|^2/\beta_g)$, where $\beta_g = \frac{1}{|\mathcal{E}|} \sum_{e \in \mathcal{E}} \|\mathbf{x}_{e_u} - \mathbf{x}_{e_v}\|^2$. For simplicity, we use the unnormalized Laplacian and so all $s(\cdot)$'s in (3) are 1. The value of $m\mu$ in (5) is always fixed at 1, and the other parameters are tuned by a small validation set. Unless otherwise specified, we use the Gaussian kernel $\exp(-\|\mathbf{x} - \mathbf{z}\|^2/\beta)$, with $\beta = \frac{1}{m} \sum_{i=1}^{m} \|\mathbf{x}_i - \bar{\mathbf{x}}\|^2$. For comparison, we also run the LapSVM[7] and another LapSVM implementation based on the reduced SVM [8] (Section 3.5). All the experiments are performed on a 3.2GHz Pentium–4 PC with 1GB RAM.

Table 1: A summary of the data sets used.

| data set | #attrib | class | #training patns labeled | #training patns unlabeled | #test patns |
|---|---|---|---|---|---|
| two-moons | 2 | + | 1 | 500,000 | 2,500 |
| | | − | 1 | 500,000 | 2,500 |
| extended USPS | 676 | + | 1 | 144,473 | 43,439 |
| | | − | 1 | 121,604 | 31,944 |
| extended MIT face | 361 | + | 5 | 408,067 | 472 |
| | | − | 5 | 481,909 | 23,573 |

### 4.1 Two-Moons Data Set

We first perform experiments on the popular two-moons data set, and use one labeled example for each class (Figure 1(a)). To better illustrate the scaling behavior, we vary the number of unlabeled patterns used for training (from $1,000$ up to a maximum of 1 million). Following [1], the width of the Gaussian kernel is set to $\beta = 0.25$. For the reduced LapSVM implementation, we fix $r = 200$.

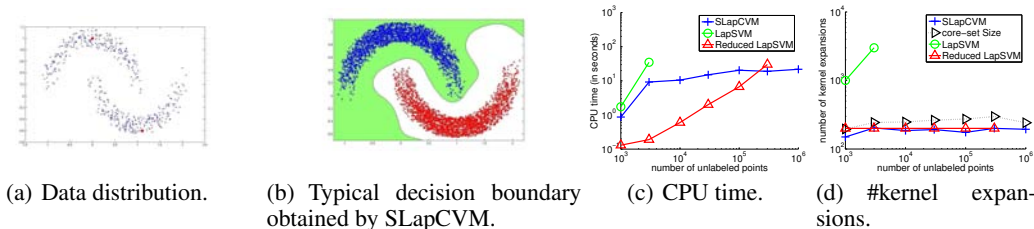

(a) Data distribution.  (b) Typical decision boundary obtained by SLapCVM.  (c) CPU time.  (d) #kernel expansions.

Figure 1: Results on the two-moons data set (some abscissas and ordinates are in log scale). The two labeled examples are labeled in red in Figure 1(a).

Results are shown in Figure 1. Both the LapSVM and SLapCVM always attain 100% accuracy on the test set, even with only two labeled examples (Figure 1(b)). However, SLapCVM is faster than LapSVM (Figure 1(c)). Moreover, as mentioned in Section 2.1, the LapSVM solution is non-sparse

and all the labeled and unlabeled examples are involved in the solution (Figure 1(d))). On the other hand, SLapCVM uses only a small fraction of the examples. As can be seen from Figures 1(c) and 1(d), both the time and space required by the SLapCVM are almost constant, even when the unlabeled data set gets very large. The reduced LapSVM, though also fast, is slightly inferior to both the SLapCVM and LapSVM. Moreover, note that both the standard and reduced LapSVMs cannot be run on the full data set on our PC because of their large memory requirements.

## 4.2 Extended USPS Data Set

The second experiment is performed on the USPS data from [12]. One labeled example is randomly sampled from each class for training. To achieve comparable accuracy, we use $r = 2,000$ for the reduced LapSVM. For comparison, we also train a standard SVM with the two labeled examples.

Results are shown in Figure 2. As can be seen, the SLapCVM is again faster (Figures 2(a)) and produces a sparser solution than LapSVM (Figure 2(b)). For the SLapCVM, both the time required and number of kernel expansions involved grow only sublinearly with the number of unlabeled examples. Figure 2(c) demonstrates that semi-supervised learning (using either the LapSVMs or SLapCVM) can have much better generalization performance than supervised learning using the labeled examples only. Note that although the use of the 2-norm error in SLapCVM could in theory be less robust than the use of the 1-norm error in LapSVM, the SLapCVM solution is indeed always more accurate than that of LapSVM. On the other hand, the reduced LapSVM has comparable speed with the SLapCVM, but its performance is inferior and cannot handle large data sets.

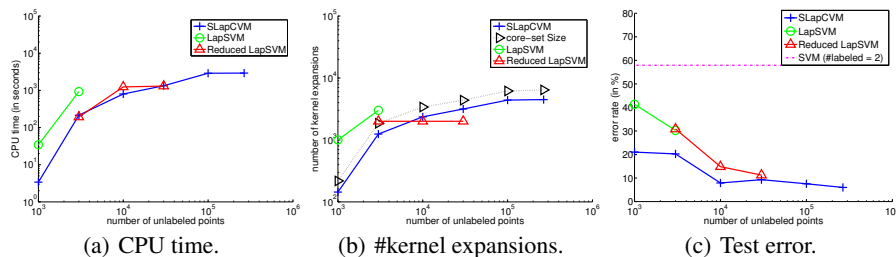

      (a) CPU time.         (b) #kernel expansions.         (c) Test error.

Figure 2: Results on the extended USPS data set (some abscissas and ordinates are in log scale).

## 4.3 Extended MIT Face Data Set

In this section, we perform face detection using the extended MIT face database in [12]. Five labeled example are randomly sampled from each class and used in training. Because of the imbalanced nature of the test set (Table 1), the classification error is inappropriate for performance evaluation here. Instead, we will use the area under the ROC curve (AUC) and the balanced loss $1 - (\text{TP} + \text{TN})/2$, where TP and TN are the true positive and true negative rates respectively. Here, faces are treated as positives while non-faces as negatives. For the reduced LapSVM, we again use $r = 2,000$. For comparison, we also train two SVMs: one uses the 10 labeled examples only while the other uses all the labeled examples (a total of 889,986) in the original training set of [12].

Figure 3 shows the results. Again, the SLapCVM is faster and produces a sparser solution than LapSVM. Note that the SLapCVM, using only 10 labeled examples, can attain comparable AUC and even better balanced loss than the SVM trained on the original, massive training set (Figures 3(c) and 3(d)). This clearly demonstrates the usefulness of semi-supervised learning when a large amount of unlabeled data can be utilized. On the other hand, note that the LapSVM again cannot be run with more than 3,000 unlabeled examples on our PC because of its high space requirement. The reduced LapSVM performs very poorly here, possibly because this data set is highly imbalanced.

## 5 Conclusion

In this paper, we addressed two issues associated with the Laplacian SVM: 1) How to obtain a sparse solution for fast testing? 2) How to handle data sets with millions of unlabeled examples? For the

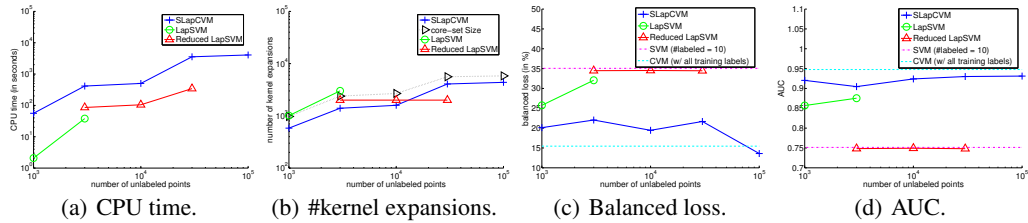

| (a) CPU time. | (b) #kernel expansions. | (c) Balanced loss. | (d) AUC. |

Figure 3: Results on the extended MIT face data (some abscissas and ordinates are in log scale).

first issue, we introduce a sparsified manifold regularizer based on the $\epsilon$-insensitive loss. For the second issue, we integrate manifold regularization with the CVM. The resultant algorithm has low time and space complexities. Moreover, by avoiding the underlying matrix inversion in the original LapSVM, a sparse solution can also be recovered. Experiments on a number of massive data sets show that the SLapCVM is much faster than the LapSVM. Moreover, while the LapSVM can only handle several thousand unlabeled examples, the SLapCVM can handle one million unlabeled examples on the same machine. On one data set, this produces comparable or even better performance than the (supervised) CVM trained on 900K labeled examples. This clearly demonstrates the usefulness of semi-supervised learning when a large amount of unlabeled data can be utilized.

## Footnotes

[1]When the set of labeled and unlabeled data is small, a function that is smooth on this small set may not be interesting. However, this is not an issue here as our focus is on massive data sets.

[2]To avoid confusion with the $\epsilon$ in the $(1 + \epsilon)$-approximation, we add a bar to the $\varepsilon$ here.

[3]For simplicity, here we have only considered the case where $f$ does not have a bias. In the presence of a bias, it can be easily shown that $\mathbf{K}$ (in the expression of $\tilde{\mathbf{K}}$) has to be replaced by $\mathbf{K} + \mathbf{1}\mathbf{1}'$.

[4]By convention, column generation takes the optimization problem to be solved as the primal. Hence, in this section, we also regard the QP to be solved as CVM's primal, and the MEB problem as its dual. Note that each dual variable then corresponds to a training pattern.

[5]Another difference is that an entire column is added at each iteration of column generation. However, in CVM, the dual variable added is just a pattern and the extra space required for the QP is much smaller. Besides, there are other implementation tricks (such as probabilistic speedup) that further improves the speed of CVM.

[6]Both the USPS and MIT face data sets are downloaded from http://www.cs.ust.hk/~ivor/cvm.html.

[7]http://manifold.cs.uchicago.edu/manifold_regularization/.

# References

[1] M. Belkin, P. Niyogi, and V. Sindhwani. Manifold regularization: A geometric framework for learning from labeled and unlabeled examples. *Journal of Machine Learning Research*, 7:2399–2434, 2006.

[2] O. Chapelle, B. Schölkopf, and A. Zien. *Semi-Supervised Learning*. MIT Press, Cambridge, MA, USA, 2006.

[3] O. Delalleau, Y. Bengio, and N. L. Roux. Efficient non-parametric function induction in semi-supervised learning. In *Proceedings of the Tenth International Workshop on Artificial Intelligence and Statistics*, Barbados, January 2005.

[4] G. Desaulniers, J. Desrosiers, and M.M. Solomon. *Column Generation*. Springer, 2005.

[5] J. Garcke and M. Griebel. Semi-supervised learning with sparse grids. In *Proceedings of the ICML Workshop on Learning with Partially Classified Training Data*, Bonn, Germany, August 2005.

[6] T. Gärtner, Q.V. Le, S. Burton, A. Smola, and S.V.N. Vishwanathan. Large-scale multiclass transduction. In Y. Weiss, B. Schölkopf, and J. Platt, editors, *Advances in Neural Information Processing Systems 18*. MIT Press, Cambridge, MA, 2006.

[7] F. Girosi. An equivalence between sparse approximation and support vector machines. *Neural Computation*, 10(6):1455–1480, 1998.

[8] Y.-J. Lee and O.L. Mangasarian. RSVM: Reduced support vector machines. In *Proceeding of the First SIAM International Conference on Data Mining*, 2001.

[9] V. Sindhwani, M. Belkin, and P. Niyogi. The geometric basis of semi-supervised learning. In *Semi-supervised Learning*. MIT Press, 2005.

[10] V. Sindhwani, P. Niyogi, and M. Belkin. Beyond the point cloud: from transductive to semi-supervised learning. In *Proceedings of the Twenty-Second International Conference on Machine Learning*, pages 825–832, Bonn, Germany, August 2005.

[11] R. Tibshirani. Regression shrinkage and selection via the Lasso. *Journal of the Royal Statistical Society: Series B*, 58:267–288, 1996.

[12] I. W. Tsang, J. T. Kwok, and P.-M. Cheung. Core vector machines: Fast SVM training on very large data sets. *Journal of Machine Learning Research*, 6:363–392, 2005.

[13] I. W. Tsang, J. T. Kwok, and K. T. Lai. Core vector regression for very large regression problems. In *Proceedings of the Twenty-Second International Conference on Machine Learning*, pages 913–920, Bonn, Germany, August 2005.

[14] X. Zhu. Semi-supervised learning literature survey. Technical Report 1530, Department of Computer Sciences, University of Wisconsin - Madison, 2005.

[15] X. Zhu and J. Lafferty. Harmonic mixtures: Combining mixture models and graph-based methods. In *Proceedings of the Twenty-Second International Conference on Machine Learning*, Bonn, Germany, August 2005.
